# Note on Learning Rate Schedules for Stochastic Optimization

**Christian Darken and John Moody**
Yale University
P.O. Box 2158 Yale Station
New Haven, CT 06520
Email: moody@cs.yale.edu

## Abstract

We present and compare learning rate schedules for stochastic gradient descent, a general algorithm which includes LMS, on-line backpropagation and k-means clustering as special cases. We introduce "search-then-converge" type schedules which outperform the classical constant and "running average" $(1/t)$ schedules both in speed of convergence and quality of solution.

## 1  Introduction: Stochastic Gradient Descent

The optimization task is to find a parameter vector $W$ which minimizes a function $G(W)$. In the context of learning systems typically $G(W) \equiv \mathcal{E}_X E(W, X)$, i.e. $G$ is the average of an objective function over the exemplars, labeled $E$ and $X$ respectively. The stochastic gradient descent algorithm is

$$\Delta W(t) = -\eta(t)\nabla_W E(W(t), X(t)).$$

where $t$ is the "time", and $X(t)$ is the most recent independently-chosen random exemplar. For comparison, the deterministic gradient descent algorithm is

$$\Delta W(t) = -\eta(t)\nabla_W \mathcal{E}_X E(W(t), X).$$

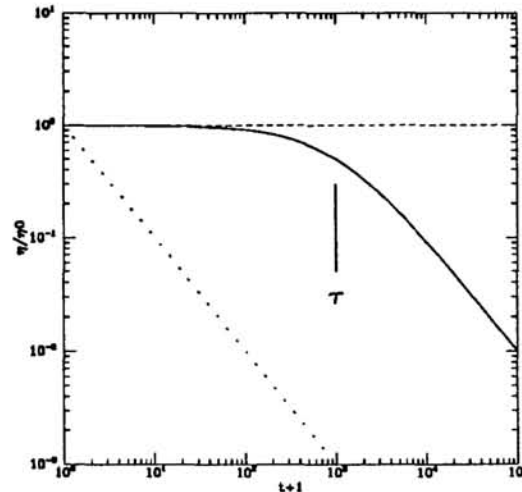

Figure 1: Comparison of the shapes of the schedules. Dashed line = constant, Solid line = search-then-converge, Dotted line = "running-average"

While on average the stochastic step is equal to the deterministic step, for any particular exemplar $X(t)$ the stochastic step may be in any direction, even uphill in $\mathcal{E}_X E(W(t), X)$. Despite its noisiness, the stochastic algorithm may be preferable when the exemplar set is large, making the average over exemplars expensive to compute.

The issue addressed by this paper is: which function should one choose for $\eta(t)$ (the learning rate schedule) in order to obtain fast convergence to a good local minimum? The schedules compared in this paper are the following (Fig. 1):

- **Constant:** $\eta(t) = \eta_0$

- **"Running Average":** $\eta(t) = \eta_0/(1 + t)$

- **Search-Then-Converge:** $\eta(t) = \eta_0/(1 + t/\tau)$

"Search-then-converge" is the name of a novel class of schedules which we introducein this paper. The specific equation above is merely one member of this class and was chosen for comparison because it is the simplest member of that class. We find that the new schedules typically outperform the classical constant and running average schedules. Furthermore the new schedules are capable of attaining the optimal asymptotic convergence rate for any objective function and exemplar distribution. The classical schedules cannot.

Adaptive schedules are beyond the scope of this short paper (see however Darken and Moody, 1991). Nonetheless, all of the adaptive schedules in the literature of which we are aware are either second order, and thus too expensive to compute for large numbers of parameters, or make no claim to asymptotic optimality.

## 2  Example Task: K-Means Clustering

As our sample gradient-descent task we choose a k-means clustering problem. Clustering is a good sample problem to study, both for its inherent usefulness and its illustrative qualities. Under the name of vector-quantization, clustering is an important technique for signal compression in communications engineering. In the machine learning field, clustering has been used as a front-end for function learning and speech recognition systems. Clustering also has many features to recommend it as an illustrative stochastic optimization problem. The adaptive law is very simple, and there are often many local minima even for small problems. Most significantly however, if the means live in a low dimensional space, visualization of the parameter vector is simple: it has the interpretation of being a set of low-dimensional points which can be easily plotted and understood.

The k-means task is to locate $k$ points (called "means") to minimize the expected distance between a new random exemplar and the nearest mean to that exemplar. Thus, the function being minimized in k-means is $\mathcal{E}_X \|X - M_{nrst}\|^2$, where $M_{nrst}$ is the nearest mean to exemplar $X$. An equivalent form is $\int dX P(X) \sum_{\alpha=1}^{k} I_\alpha(X) \|X - M_\alpha\|^2$, where $P(X)$ is the density of the exemplar distribution and $I_\alpha(X)$ is the indicator function of the Veronois region corresponding to the $\alpha$th mean. The stochastic gradient descent algorithm for this function is

$$\Delta M_{nrst}(t) = -\eta(t_{nrst})[M_{nrst}(t) - X(t)],$$

i.e. the nearest mean to the latest exemplar moves directly towards the exemplar a fractional distance $\eta(t_{nrst})$. In a slight generalization from the stochastic gradient descent algorithm above, $t_{nrst}$ is the total number of exemplars (including the current one) which have been assigned to mean $M_{nrst}$.

As a specific example problem to compare various schedules across, we take $k = 9$ (9 means) and $X$ uniformly distributed over the unit square. Although this would appear to be a simple problem, it has several observed local minima. The global minimum is where the means are located at the centers of a uniform 3x3 grid over the square. Simulation results are presented in figures 2 and 3.

## 3  Constant Schedule

A constant learning rate has been the traditional choice for LMS and backpropagation. However, a constant rate generally does not allow the parameter vector (the "means" in the case of clustering) to converge. Instead, the parameters hover around a minimum at an average distance proportional to $\eta$ and to a variance which depends on the objective function and the exemplar set. Since the statistics of the exemplars are generally assumed to be unknown, this residual misadjustment cannot be predicted. The resulting degradation of other measures of system performance, mean squared classification error for instance, is still more difficult to predict. Thus the study of how to make the parameters converge is of significant practical interest.

Current practice for backpropagation, when large misadjustment is suspected, is to restart learning with a smaller $\eta$. Shrinking $\eta$ does result in less residual misadjustment, but at the same time the speed of convergence drops. In our example

clustering problem, a new phenomenon appears as $\eta$ drops—metastable local minima. Here the parameter vector hovers around a relatively poor solution for a very long time before slowly transiting to a better one.

## 4   Running Average Schedule

The running average schedule $(\eta(t) = \eta_0/(1 + t))$ is the staple of the stochastic approximation literature (Robbins and Monro, 1951) and of k-means clustering (with $\eta_0 = 1$) (MacQueen, 1967). This schedule is optimal for $k = 1$ (1 mean), but performs very poorly for moderate to large $k$ (like our example problem with 9 means). From the example run (Fig. 2A), it is clear that $\eta$ must decrease more slowly in order for a good solution to be reached. Still, an advantage of this schedule is that the parameter vector has been proven to converge to a local minimum (MacQueen, 1967). We would like a class of schedules which is guaranteed to converge, and yet converges as quickly as possible.

## 5   Stochastic Approximation Theory

In the stochastic approximation literature, which has grown steadily since it began in 1951 with the Robbins and Monro paper, we find conditions on the learning rate to ensure convergence with optimal speed [1].

From (Ljung, 1977), we find that $\eta(t) \to At^{-p}$ asymptotically for any $1 \geq p > 0$, is sufficient to guarantee convergence. Power law schedules may work quite well in practice (Darken and Moody, 1990), however from (Goldstein, 1987) we find that in order to converge at an optimal rate, we must have $\eta(t) \to c/t$ asymptotically, for $c$ greater than some threshold which depends on the objective function and exemplars [2]. When the optimal convergence rate is achieved, $\|W - W^*\|^2$ goes like $1/t$.

The running average schedule goes as $\eta_0/t$ asymptotically. Unfortunately, the convergence rate of the running average schedule often cannot be improved by enlarging $\eta_0$, because the resulting instability for small $t$ can outweigh the improvements in asymptotic convergence rate.

## 6   Search-Then-Converge Schedules

We now introduce a new class of schedules which are guaranteed to converge and furthermore, can achieve the optimal $1/t$ convergence rate without stability problems. These schedules are characterized by the following features. The learning rate stays high for a "search time" $\tau$ in which it is hoped that the parameters will find and hover about a good minimum. Then, for times greater than $\tau$, the learning rate decreases as $c/t$, and the parameters converge.

We choose the simplest of this class of schedules for study, the "short-term linear" schedule $(\eta(t) = \eta_0/(1 + t/\tau))$, so called because the learning rate decreases linearly during the search phase. This schedule has $c \equiv \tau\eta_0$ and reduces to the running average schedule for $\tau = 1$.

# 7    Conclusions

We have introduced the new class of "search-then-converge" learning rate schedules. Stochastic approximation theory indicates that for large enough $\tau$, these schedules can achieve optimally fast asymptotic convergence for any exemplar distribution and objective function. Neither constant nor "running average" $(1/t)$ schedules can achieve this. Empirical measurements on k-means clustering tasks are consistent with this expectation. Furthermore asymptotic conditions obtain surprisingly quickly. Additionally, the search-then-converge schedule improves the observed likelihood of escaping bad local minima.

As implied above, k-means clustering is merely one example of a stochastic gradient descent algorithm. LMS and on-line backpropagation are others of great interest to the learning systems community. Due to space limitations, experiments in these settings will be published elsewhere (Darken and Moody, 1991). Preliminary experiments seem to confirm the generality of the above conclusions.

Extensions to this work in progress includes application to algorithms more sophisticated than simple gradient descent, and adaptive search-then-converge algorithms which automatically determine the search time.

## Acknowledgements

The authors wish to thank Hal White for useful conversations and Jon Kauffman for developing the animator which was used to produce figure 2. This work was supported by ONR Grant N00014-89-J-1228 and AFOSR Grant 89-0478.

## Footnotes

[1]The cited theory generally does not directly apply to the full nonlinear setting of interest in much practical work. For more details on the relation of the theory to practical applications and a complete quantitative theory of asymptotic misadjustment, see (Darken and Moody, 1991).

[2]This choice of asymptotic $\eta$ satisfies the necessary conditions given in (White, 1989).

## References

C. Darken and J. Moody. (1990) Fast Adaptive K-Means Clustering: Some Empirical Results. In *International Joint Conference on Neural Networks 1990*, 2:233-238. IEEE Neural Networks Council.

C. Darken and J. Moody. (1991) Learning Rate Schedules for Stochastic Optimization. In preparation.

L. Goldstein. (1987) Mean square optimality in the continuous time Robbins Monro procedure. Technical Report DRB-306. Department of Mathematics, University of Southern California.

L. Ljung. (1977) Analysis of Recursive Stochastic Algorithms. *IEEE Trans. on Automatic Control.* **AC-22**(4):551-575.

J. MacQueen. (1967) Some methods for classification and analysis of multivariate observations. In *Proc. 5th Berkeley Symp. Math. Stat. Prob.* **3**:281.

H. Robbins and S. Monro. (1951) A Stochastic Approximation Method. *Ann. Math. Stat.* **22**:400-407.

H. White. (1989) Learning in Artificial Neural Networks: A Statistical Perspective. *Neural Computation.* 1:425-464.

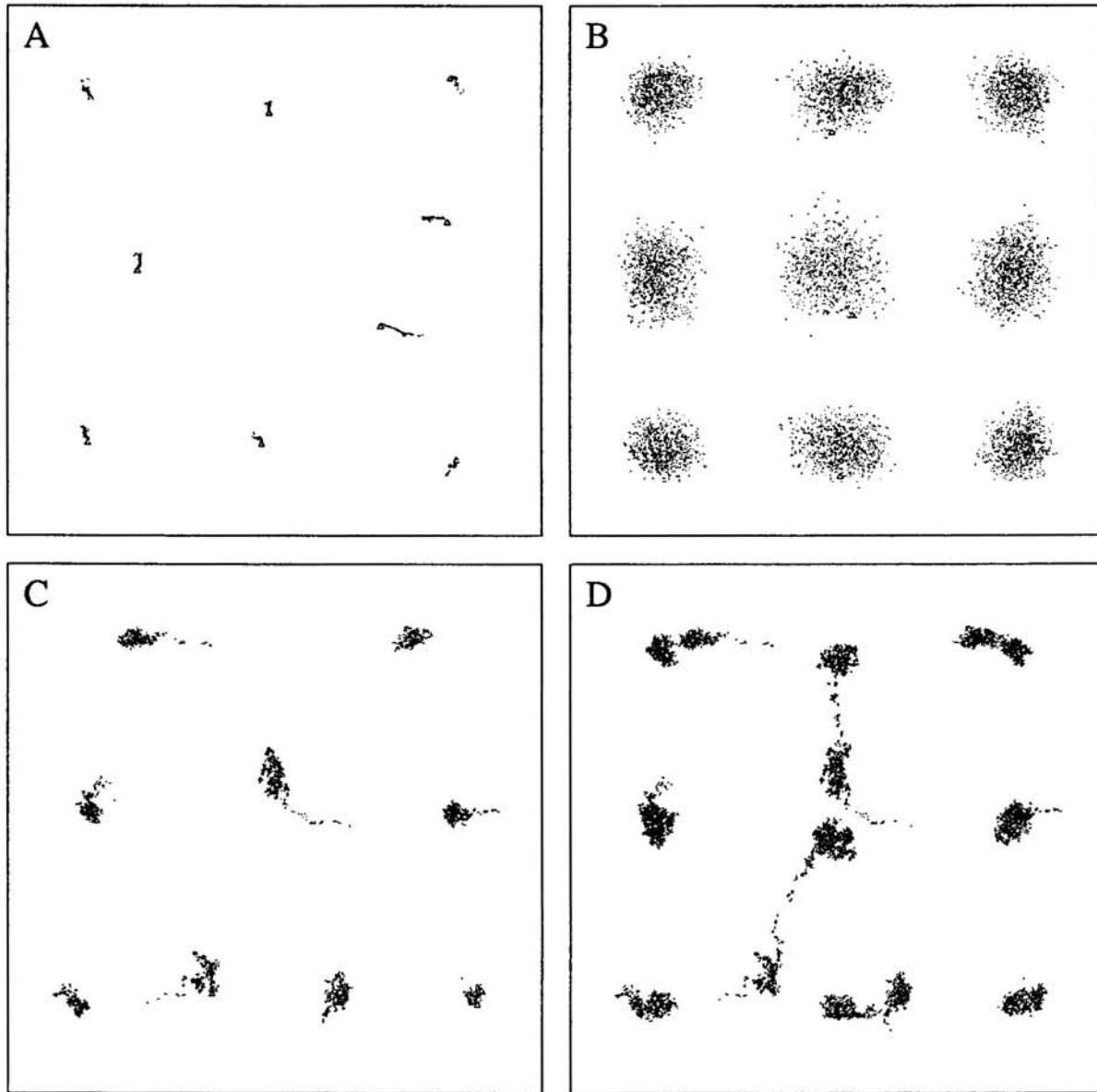

**Figure 2:** Example runs with classical schedules on 9-means clustering task. Exemplars are uniformly distributed over the square. Dots indicate previous locations of the means. The triangles (barely visible) are the final locations of the means. (A) "Running average" schedule ($\eta = 1/(1 + t)$), 100k exemplars. Means are far from any minimum and progressing very slowly. (B) Large constant schedule ($\eta=0.1$), 100k exemplars. Means hover around global minimum at large average distance. (C) Small constant schedule ($\eta=0.01$), 50k exemplars. Means stuck in metastable local minimum. (D) Small constant schedule ($\eta=0.01$), 100k exemplars (later in the run pictured in C). Means tunnel out of local minimum and hover around global minimum.

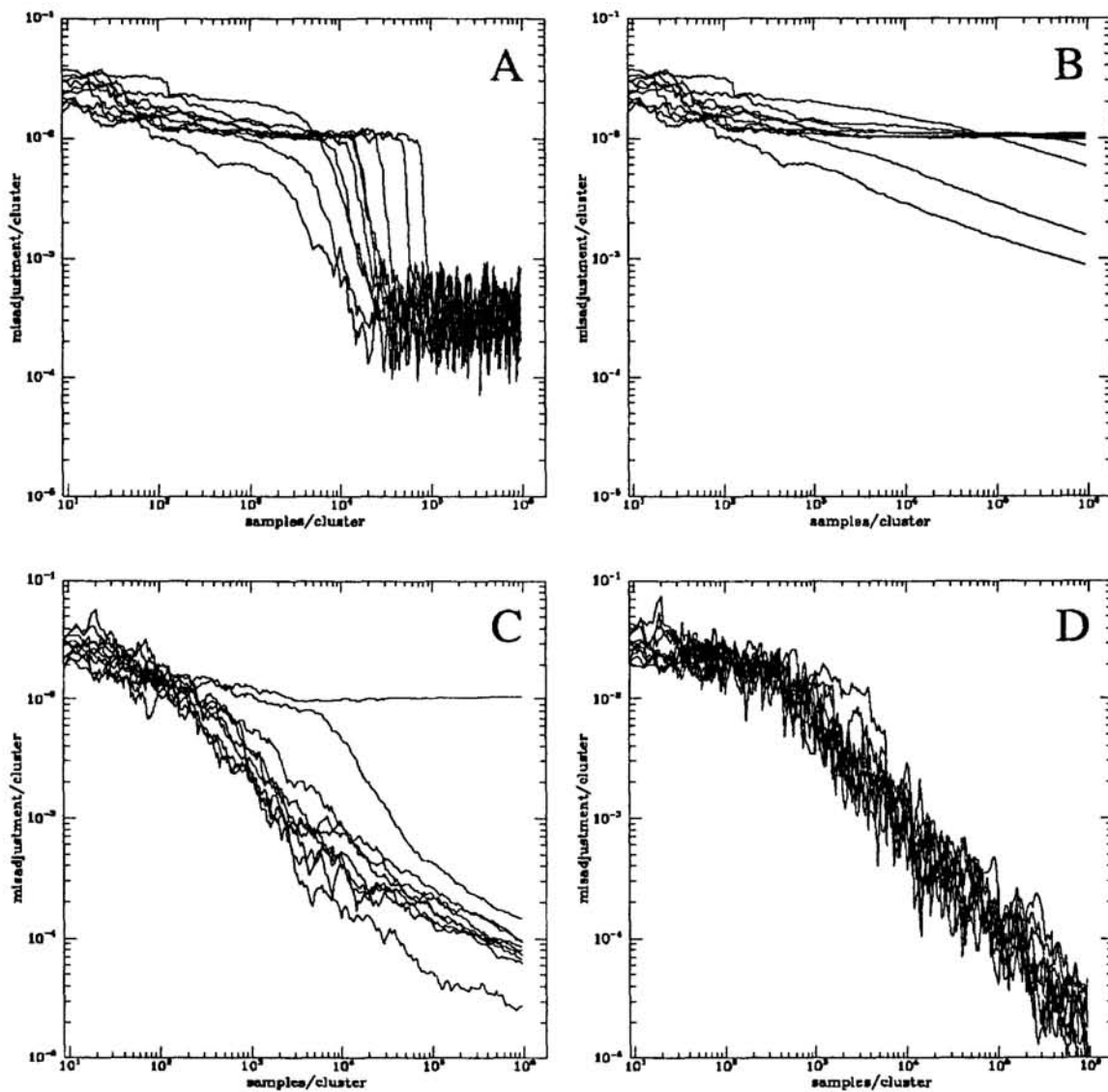

**Figure 3:** Comparison of 10 runs over the various schedules on the 9-means cluster-
ing task (as described under Fig. 1). The exemplars are the same for each schedule.
Misadjustment is defined as $||W - W^{best}||^2$. (A) Small constant schedule ($\eta$=0.01).
Note the well-defined transitions out of metastable local minima and large misad-
justment late in the runs. (B) "Running average" schedule ($\eta = 1/(1 + t)$). 6
out of 10 runs stick in a local minimum. The others slowly head for the global
minimum. (C) Search-then-converge schedule ($\eta = 1/(1 + t/4)$). All but one run
head for global minimum, but at a suboptimal rate (asymptotic slope less than -1).
(D) Search-then-converge schedule ($\eta = 1/(1 + t/32)$). All runs head for global
minimum at optimally quick rate (asymptotic slope of -1).